# Online Learning in the Manifold of
# Low-Rank Matrices

**Uri Shalit**[*]**, Daphna Weinshall**
Computer Science Dept. and ICNC
The Hebrew University of Jerusalem
uri.shalit@mail.huji.ac.il
daphna@cs.huji.ac.il

**Gal Chechik**
Google Research and
The Gonda Brain Research Center
Bar Ilan University
gal@google.com

## Abstract

When learning models that are represented in matrix forms, enforcing a low-rank constraint can dramatically improve the memory and run time complexity, while providing a natural regularization of the model. However, naive approaches for minimizing functions over the set of low-rank matrices are either prohibitively time consuming (repeated singular value decomposition of the matrix) or numerically unstable (optimizing a factored representation of the low rank matrix). We build on recent advances in optimization over manifolds, and describe an iterative online learning procedure, consisting of a gradient step, followed by a second-order *retraction* back to the manifold. While the ideal retraction is hard to compute, and so is the projection operator that approximates it, we describe another second-order retraction that can be computed efficiently, with run time and memory complexity of $O\left((n+m)k\right)$ for a rank-$k$ matrix of dimension $m \times n$, given rank-one gradients. We use this algorithm, LORETA, to learn a matrix-form similarity measure over pairs of documents represented as high dimensional vectors. LORETA improves the mean average precision over a passive- aggressive approach in a factorized model, and also improves over a full model trained over pre-selected features using the same memory requirements. LORETA also showed consistent improvement over standard methods in a large (1600 classes) multi-label image classification task.

## 1   Introduction

Many learning problems involve models represented in matrix form. These include metric learning, collaborative filtering, and multi-task learning where all tasks operate over the same set of features. In many of these models, a natural way to regularize the model is to limit the rank of the corresponding matrix. In metric learning, a low rank constraint allows to learn a low dimensional representation of the data in a discriminative way. In multi-task problems, low rank constraints provide a way to tie together different tasks. In all cases, low-rank matrices can be represented in a factorized form that dramatically reduces the memory and run-time complexity of learning and inference with that model. Low-rank matrix models could therefore scale to handle substantially many more features and classes than with full rank dense matrices.

As with many other problems, the rank constraint is non-convex, and in the general case, minimizing a convex function subject to a rank constraint is NP-hard [1] [1]. As a result, two main approaches have been commonly used. Sometimes, a matrix $W \in \mathbb{R}^{n \times m}$ of rank $k$ is represented as a product of two low dimension matrices $W = AB^T, A \in \mathbb{R}^{n \times k}, B \in \mathbb{R}^{m \times k}$ and simple gradient descent techniques are applied to each of the product terms separately [3]. Second, projected gradient algorithms can be applied by repeatedly taking a gradient step and projecting back to the manifold of low-rank matrices. Unfortunately, computing the projection to that manifold becomes prohibitively costly for large matrices and cannot be computed after every gradient step.

---

[*]also at the Gonda Brain Research Center, Bar Ilan University
[1]Some special cases are solvable (notably, PCA), relying mainly on singular value decomposition [2] and semi-definite programming techniques. These methods scale poorly to large scale tasks.

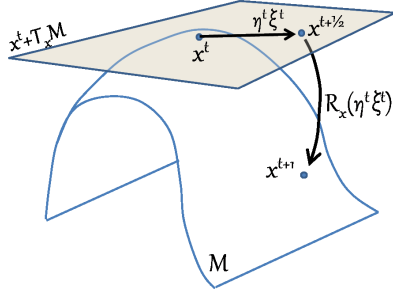

Figure 1: A two step procedure for computing a retracted gradient. The first step computes the Riemannian gradient $\xi$ (the projection of the gradient onto the tangent space $T_x\mathcal{M}_k^{n,m}$), yielding $x^{t+\frac{1}{2}} = x^t + \eta^t\nabla\mathcal{L}(x^t)$. The second step computes the retraction onto the manifold $x^{t+1} = R_x(\xi^t)$.

In this paper we propose new algorithms for online learning on the manifold of low-rank matrices, which are based on an operation called *retraction*. Retractions are operators that map from a vector space that is tangent to the manifold, into the manifold. They include the projection operator as a special case, but also include other retractions that can be computed dramatically more efficiently. We use second order retractions to develop LORETA – an online algorithm for learning low rank matrices. It has a memory and run time complexity of $O\left((n+m)k\right)$ when the gradients have rank one, a case which is relevant to numerous online learning problems as we show below.

We test Loreta in two different domains and learning tasks. First, we learn a bilinear similarity measure among pairs of text documents, where the number of features (text terms) representing each document could become very large. Loreta performed better than other techniques that operate on a factorized model, and also improves retrieval precision by 33% as compared with training a full rank model over pre-selected most informative features, using comparable memory footprint. Second, we applied Loreta to image multi-label ranking, a problem in which the number of classes could grow to millions. Loreta significantly improved over full rank models, using a fraction of the memory required. These two experiments suggest that low-rank optimization could become very useful for learning in high-dimensional problems.

This paper is organized as follows. We start with an introduction to optimization on manifolds, describing the notion of retractions. We then derive our low-rank online learning algorithm, and test it in two applications: learning similarity of text documents, and multi-label ranking for images.

## 2 Optimization on Riemannian manifolds

The field of numerical optimization on smooth manifolds has advanced significantly in the past few years. We start with a short introduction to embedded manifolds, which are the focus of this paper. An *embedded manifold* is a smooth subset of an ambient space $\mathbb{R}^n$. For instance the set $\{\mathbf{x} : ||x||_2 = 1, \mathbf{x} \in \mathbb{R}^n\}$, the unit sphere, is an $n-1$ dimensional manifold embedded in $n$-dimensional space $\mathbb{R}^n$. Here we focus on the manifold of *low-rank* matrices, namely, the set of $n \times m$ matrices of rank $k$ where $k < m, n$. It is an $(n+m)k - k^2$ dimensional manifold embedded in $\mathbb{R}^{n \times m}$, which we denote $\mathcal{M}_k^{n,m}$. Embedded manifolds inherit many properties from the ambient space, a fact which simplifies their analysis. For example, the Riemannian metric for embedded manifolds is simply the Euclidean metric restricted to the manifold.

Motivated by online learning, we focus here on developing a stochastic gradient descent procedure to minimize a loss function $\mathcal{L}$ over the manifold of low-rank matrices $\mathcal{M}_k^{n,m}$,

$$\min_x \quad \mathcal{L}(x) \qquad \text{s.t.} \quad x \in \mathcal{M}_k^{n,m} \quad . \tag{1}$$

To illustrate the challenge in this problem, consider a simple stochastic gradient descent algorithm (Fig. 1). At every step $t$ of the algorithm, a gradient step update takes $x^{t+\frac{1}{2}}$ outside of the manifold $\mathcal{M}$ and has to be mapped back onto the manifold. The most common mapping operation is the *projection* operation, which, given a point $x^{t+\frac{1}{2}}$ outside the manifold, would find the closest point in $\mathcal{M}$. Unfortunately, the projection operation is very expensive to compute for the manifold of low rank matrices, since it basically involves a singular value decomposition. Here we describe a wider class of operations called *retractions*, that serve a similar purpose: they find a point on the manifold that is in the direction of the gradient. Importantly, we describe a specific retraction that can be computed efficiently. Its runtime complexity depends on 4 quantities: the model matrix dimensions $m$ and $n$; its rank $k$; and the rank of the gradient matrix, $r$. The overall complexity is $O\left((n+m)(k+r)^2\right)$, and $O\left((n+m)k\right)$ for rank-one gradients, which are a very common case.

To explain how retractions are computed, we first describe the notion of a *tangent space* and the *Riemannian gradient* of a function on a manifold.

**Riemannian gradient and the tangent space**

Each point $x$ in an embedded manifold $\mathcal{M}$ has a tangent space associated with it, denoted $T_\mathbf{x}\mathcal{M}$ (see Fig. 1). The tangent space is a vector space of the same dimension as the manifold that can be identified in a natural way with a linear subspace of the ambient space. It is usually simple to compute the linear projection $P_x$ of any point in the ambient space onto the tangent space $T_\mathbf{x}\mathcal{M}$.

Given a manifold $\mathcal{M}$ and a differentiable function $\mathcal{L} : \mathcal{M} \to \mathbb{R}$, the *Riemannian gradient* $\nabla\mathcal{L}(x)$ of $\mathcal{L}$ on $\mathcal{M}$ at a point $\mathbf{x}$ is a vector in the tangent space $T_\mathbf{x}\mathcal{M}$. A very useful property of embedded manifolds is the following: given a differentiable function $f$ defined on the ambient space (and thus on the manifold), the Riemannian gradient of $f$ at point $x$ is simply the linear projection $P_x$ of the ordinary gradient of $f$ onto the tangent space $T_x\mathcal{M}$. An important consequence follows in case the manifold represents the set of points obeying a certain constraint. In this case the Riemannian gradient of $f$ is equivalent to the ordinary gradient of the $f$ minus the component which is normal to the constraint. Indeed this normal component is exactly the component which is irrelevant when performing constrained optimization.

The Riemannian gradient allows us to compute $x^{t+\frac{1}{2}} = x^t + \eta^t \nabla\mathcal{L}(x)$, for a given iterate point $x^t$ and step size $\eta^t$. We now examine how $x^{t+\frac{1}{2}}$ can be mapped back onto the manifold.

**Retractions**

Intuitively, *retractions* capture the notion of "going along a straight line" on the manifold. The mathematically ideal retraction is called the *exponential mapping*: it maps the tangent vector $\xi \in T_x\mathcal{M}$ to a point along a geodesic curve which goes through $x$ in the direction of $\xi$. Unfortunately, for many manifolds (including the low-rank manifold considered here) calculating the geodesic curve is computationally expensive. A major insight from the field of Riemannian manifold optimization is that using the exponential mapping is unnecessary since computationally cheaper retractions exist.

Formally, for a point $x$ in an embedded manifold $\mathcal{M}$, a retraction is any function $R_x : T_\mathbf{x}\mathcal{M} \to \mathcal{M}$ which satisfies the following two conditions [4]: (1) Centering: $R_x(0) = x$. (2) Local rigidity: the curve defined by $\gamma_\xi(\tau) = R_x(\tau\xi)$ satisfies $\dot{\gamma}_\xi(0) = \xi$. It can be shown that any such retraction approximates the exponential mapping to a first order [4]. *Second-order retractions*, which approximate the exponential mapping to second order around $x$, have to satisfy the following stricter conditions: $P_x\left(\frac{\mathrm{d}R_x(\tau\xi)}{\mathrm{d}\tau^2}|_{\tau=0}\right) = 0$, for all $\xi \in T_x\mathcal{M}$, where $P_x$ is the linear projection from the ambient space onto the tangent space $T_\mathbf{x}\mathcal{M}$. When viewed intrinsically, the curve $R_x(\tau\xi)$ defined by a second-order retraction has zero acceleration at point $x$, namely, its second order derivatives are all normal to the manifold. The best known example of a second-order retraction onto embedded manifolds is the projection operation [5]. Importantly, projections are viewed here as one type of a second order approximation to the exponential mapping, which can be replaced by any other second-order retractions, when computing the projection is too costly.

Given the tangent space and a retraction, we can now define a Riemannian gradient descent step for the loss $\mathcal{L}$ at point $x^t \in \mathcal{M}$:
**(1) Gradient step:** Compute $x^{t+\frac{1}{2}} = x^t + \xi^t$, with $\xi^t = \nabla\mathcal{L}(x^t) = P_{x^t}(\tilde{\nabla}\mathcal{L}(x^t))$, where $\tilde{\nabla}\mathcal{L}(x^t)$ is the ordinary gradient of $\mathcal{L}$ in the ambient space.
**(2) Retraction step:** Compute $x^{t+1} = R_{x^t}(-\eta^t\xi^t)$, where $\eta^t$ is the step size.
For a proper step size, this procedure can be proved to have local convergence for any retraction [4].

# 3 Online learning on the low rank manifold

Based on the retractions described above, we now present an online algorithm for learning low-rank matrices, by performing stochastic gradient descent on the manifold of low rank matrices. At every iteration the algorithm suffers a loss, and performs a Riemannian gradient step followed by a retraction to the manifold $\mathcal{M}_k^{n,m}$. Section 3.1 discusses general online updates. Section 3.2 discusses the very common case where the online updates induce a gradient of rank $r = 1$.

In what follows, a lowercase $x$ denotes an abstract point on the manifold, lowercase Greek letters like $\xi$ denote an abstract tangent vector, and uppercase Roman letters like $A$ denote concrete matrix

representations as kept in memory (taking $n \times m$ float numbers to store). We intermix the two notations, as in $\xi = AZ$, when the meaning is clear from the context. The set of $n \times k$ matrices of rank $k$ is denoted $\mathbb{R}_*^{n \times k}$.

## 3.1 The general LORETA algorithm

We start with a Lemma that gives a representation of the tangent space $T_{\mathbf{x}} \mathcal{M}$, extending the constructions given in [6] to the general manifold of low-rank matrices. The proof is given in the supplemental material.

**Lemma 1.** *Let $x \in \mathcal{M}_k^{n,m}$ have a (non-unique) factorization $x = AB^T$, where $A \in \mathbb{R}_*^{n \times k}$, $B \in \mathbb{R}_*^{m \times k}$. Let $A_\perp \in \mathbb{R}^{n \times (n-k)}$ and $B_\perp \in \mathbb{R}^{m \times (m-k)}$ be the orthogonal complements of $A$ and $B$ respectively, such that $A_\perp^T A = 0$, $B_\perp^T B = 0$, $A_\perp^T A_\perp = I_{n-k}$, $B_\perp^T B_\perp = I_{m-k}$. The tangent space to $\mathcal{M}_k^{n,m}$ at $x$ is:*

$$T_{\mathbf{x}} \mathcal{M} = \left\{ [A \quad A_\perp] \begin{bmatrix} M & N_1^T \\ N_2 & 0 \end{bmatrix} \begin{bmatrix} B^T \\ B_\perp^T \end{bmatrix} : M \in \mathbb{R}^{k \times k}, N_1 \in \mathbb{R}^{(m-k) \times k}, N_2 \in \mathbb{R}^{(n-k) \times k} \right\} \quad (2)$$

Let $\xi \in \mathcal{M}_k^{n,m}$ be a tangent vector to $x = AB^T$. From the characterization above it follows that $\xi$ can be decomposed in a unique manner into three orthogonal components: $\xi = \xi^S + \xi_l^P + \xi_r^P$, where $\xi^S = AMB^T$, $\xi_l^P = AN_1^T B_\perp^T$ and $\xi_r^P = A_\perp N_2 B^T$. In online learning we are repeatedly given a rank-$r$ gradient matrix $Z$, and want to compute a step on $\mathcal{M}_k^{n,m}$ in the direction of $Z$. As a first step we wish to find its projection $P_x(Z)$ onto the tangent space. Specifically, we wish to find the three matrices $M$, $N_1$ and $N_2$ such that $P_x(Z) = AMB^T + AN_1^T B_\perp^T + A_\perp N_2 B^T$. Since we assume $A$ is of full column rank, its pseudo-inverse $A^\dagger$ obeys $A^\dagger = (A^T A)^{-1} A^T$. The matrix projecting onto $A$'s columns, denoted $P_A$, is exactly equal to $AA^\dagger$. We can similarly define $P_{A_\perp}$, $P_B$ and $P_{B_\perp}$. A straightforward computation shows that for a given matrix $Z$, we have $M = A^\dagger Z B^{\dagger T}$, $N_1 = B_\perp^T Z^T A^{\dagger T}$, $N_2 = A_\perp^T Z B_\perp^T$, yielding $\xi^S = P_A Z P_B$, $\xi_l^P = P_A Z P_{B_\perp}$, $\xi_r^P = P_{A_\perp} Z P_B$.

The following theorem defines the retraction that we use. The proof is given in the supplemental material.

**Theorem 1.** *Let $x \in \mathcal{M}_k^{n,m}$, $x = AB^T$, and $x^\dagger = B^{\dagger T} A^\dagger = B(B^T B)^{-1}(A^T A)^{-1} A^T$ (this holds since we assume $A$ and $B$ are of full column rank). Let $\xi \in T_x \mathcal{M}_k^{n,m}$, $\xi = \xi^S + \xi_l^P + \xi_r^P$, as described above, and let*

$$w_1 = x + \frac{1}{2}\xi^S + \xi_r^P - \frac{1}{8}\xi^S x^\dagger \xi^S - \frac{1}{2}\xi_r^P x^\dagger \xi^S \quad , \quad (3)$$

$$w_2 = x + \frac{1}{2}\xi^S + \xi_l^P - \frac{1}{8}\xi^S x^\dagger \xi^S - \frac{1}{2}\xi^S x^\dagger \xi_l^P \quad .$$

*The mapping $R_x(\xi) = w_1 x^\dagger w_2$ is a second order retraction from a neighborhood $\Theta_x \subset T_x \mathcal{M}_k^{n,m}$ to $\mathcal{M}_k^{n,m}$.*

We now have the ingredients necessary for a Riemannian stochastic gradient descent algorithm.

---

**Algorithm 1** : Naive Riemannian stochastic gradient descent

---

**Input:** Matrices $A \in \mathbb{R}_*^{n \times k}$, $B \in \mathbb{R}_*^{m \times k}$ s.t. $x = AB^T$. Matrices $G_1 \in \mathbb{R}^{n \times r}$, $G_2 \in \mathbb{R}^{m \times r}$ s.t. $G_1 G_2^T = -\eta \xi = -\eta \tilde{\nabla} \mathcal{L}(x) \in \mathbb{R}^{n \times m}$, where $\tilde{\nabla} \mathcal{L}(x)$ is the gradient in the ambient space and $\eta > 0$ is the step size.
**Output:** Matrices $Z_1 \in \mathbb{R}_*^{n \times k}$, $Z_2 \in \mathbb{R}_*^{m \times k}$ such that $Z_1 Z_2^T = R_x(-\eta \xi)$.
**Compute:**                                                                   matrix dimension

$\quad A^\dagger = (A^T A)^{-1} A^T$, $B^\dagger = (B^T B)^{-1} B^T$                           $k \times n, \quad k \times m$

$\quad A_\perp, B_\perp =$ orthogonal complements of $A, B$                       $n \times (n-k), \quad m \times (m-k)$

$\quad M = A^\dagger G_1 G_2^T B^{\dagger T}$                                              $k \times k$

$\quad N_1 = B_\perp^T G_2 G_1^T A^{\dagger T}, \quad N_2 = A_\perp^T G_1 G_2^T B^{\dagger T}$         $(m-k) \times k, \quad (n-k) \times k$

$\quad Z_1 = \left( A \left( I_k + \frac{1}{2}M - \frac{1}{8}M^2 \right) + A_\perp N_2 \left( I_k - \frac{1}{2}M \right) \right)$     $n \times k$

$\quad Z_2 = \left( B \left( I_k + \frac{1}{2}M^T - \frac{1}{8}(M^T)^2 \right) + B_\perp N_1 \left( I_k - \frac{1}{2}M^T \right) \right)$   $m \times k$

---

Given a gradient in the ambient space $\tilde{\nabla} \mathcal{L}(x)$, we can calculate the matrices $M$, $N_1$ and $N_2$ which allow us to represent its projection onto the tangent space, and furthermore allow us to calculate the retraction. The procedure is outlined in algorithm 1, with some rearranging and term collection.

Algorithm 1 explicitly computes and stores the orthogonal complement matrices $A_\perp$ and $B_\perp$, which in the low rank case $k \ll m, n$, have size $O(mn)$ as the original $x$. To improve the memory complexity, we use the fact that the matrices $A_\perp$ and $B_\perp$ always operate with their transpose. Since they are orthogonal, the matrix $A_\perp A_\perp^T$ is a projection matrix, one which we denoted earlier by $P_{A_\perp}$, and likewise for $B_\perp$. Because of the orthogonal complementarity, these projection matrices are equal to $I_n - P_A$ and $I_m - P_B$ respectively. We use this identity to reformulate the algorithm such that only matrices of size at most $\max(n, m) \times k$ or $\max(n, m) \times r$ are kept in memory. The runtime complexity of Algorithm 2 can be easily computed based on matrix multiplications complexity, and equals $O\big((n + m)(k + r)^2\big)$.

---

**Algorithm 2** : General Riemannian stochastic gradient descent

---

**Input and Output**: As in Algorithm 1

| **Compute:** | matrix dimension |
|---|---|
| $A^\dagger = (A^T A)^{-1} A^T, \quad B^\dagger = (B^T B)^{-1} B^T$ | $k \times n, \quad k \times m$ |
| $\hat{A} = A^\dagger \cdot G_1, \quad \hat{B} = B^\dagger \cdot G_2$ | $k \times r, \quad k \times r$ |
| $ProjAG = A \cdot \hat{A}$ | $n \times r$ |
| $Q = \hat{B}^T \cdot \hat{A}$ | $r \times r$ |
| $A^\Delta = -\frac{1}{2} ProjAG + \frac{3}{8} ProjAG \cdot Q + G_1 - \frac{1}{2} G_1 \cdot Q$ | $n \times r$ |
| $Z_1 = A + A^\Delta \cdot \hat{B}^T$ | $n \times k$ |
| $GBproj = \big(G_2^T B\big) \cdot B^\dagger$ | $r \times m$ |
| $B^\Delta = -\frac{1}{2} GBproj + \frac{3}{8} Q \cdot GBproj + G_2^T - \frac{1}{2} Q \cdot G_2^T$ | $r \times m$ |
| $Z_2^T = B^T + \hat{A} \cdot B^\Delta$ | $k \times m$ |

---

**Algorithm 3** , **Loreta-1:** Rank-one Riemannian stochastic gradient descent

---

**Input:** Matrices $A \in \mathbb{R}_*^{n \times k}$, $B \in \mathbb{R}_*^{m \times k}$ s.t. $x = AB^T$. Matrices $A^\dagger$ and $B^\dagger$, the pseudo-inverses of $A$ and $B$ respectively. Vectors $G_1 \in \mathbb{R}^{n \times 1}$, $G_2 \in \mathbb{R}^{m \times 1}$ s.t. $G_1 G_2^T = -\eta \xi = -\eta \tilde{\nabla} \mathcal{L}(x) \in \mathbb{R}^{n \times m}$, where $\tilde{\nabla} \mathcal{L}(x)$ is the gradient in the ambient space and $\eta > 0$ is the step size.
**Output:** Matrices $Z_1 \in \mathbb{R}_*^{n \times k}$, $Z_2 \in \mathbb{R}_*^{m \times k}$ s.t. $Z_1 Z_2^T = R_x(-\eta \xi)$. Matrices $Z_1^\dagger$ and $Z_2^\dagger$, the pseudo-inverses of $Z_1$ and $Z_2$ respectively.

| **Compute:** | matrix dimension |
|---|---|
| $\hat{A} = A^\dagger \cdot G_1, \hat{B} = B^\dagger \cdot G_2$ | $k \times 1$ |
| $ProjAG = A \cdot \hat{A}$ | $n \times 1$ |
| $Q = \hat{B}^T \cdot \hat{A}$ | $1 \times 1$ |
| $A^\Delta = ProjAG \left(-\frac{1}{2} + \frac{3}{8} Q\right) + G_1 (1 - \frac{1}{2} Q)$ | $n \times 1$ |
| $Z_1 = A + A^\Delta \cdot \hat{B}^T$ | $n \times k$ |
| $GBproj = \big(G_2^T B\big) \cdot B^\dagger$ | $1 \times m$ |
| $B^\Delta = GBproj \left(-\frac{1}{2} + \frac{3}{8} Q\right) + G_2^T (1 - \frac{1}{2} Q)$ | $1 \times m$ |
| $Z_2^T = B^T + \hat{A} \cdot B^\Delta$ | $k \times m$ |
| $Z_1^\dagger = rank\_one\_pseudoinverse\_update(A, A^\dagger, A^\Delta, \hat{B})$ | $k \times n$ |
| $Z_2^\dagger = rank\_one\_pseudoinverse\_update(B, B^\dagger, B^\Delta, \hat{A})$ | $k \times m$ |

### 3.2 LORETA with rank-one gradients

In many learning problems, the gradient matrix required for a gradient step update has a rank of one. This is the case for example, when the matrix model $W$ acts as a bilinear form on two vectors, $p$ and $q$, and the loss is a linear function of $\mathbf{p}^T W \mathbf{q}$ (as in [7, 8], and Sec. 5.1). In that case, the gradient is the rank-one, outer product matrix $\mathbf{p q}^T$. As another example, consider the case of multitask learning, where the matrix model $W$ operates on a vector input $\mathbf{p}$, and the loss is the squared loss between the multiple predictions $W \mathbf{p}$ and the true labels $q$: $\|W \mathbf{p} - \mathbf{q}\|^2$. The gradient of the loss is $(W \mathbf{p} - \mathbf{q}) \mathbf{p}^T$, which is again a rank-one matrix. We now show how to reduce the complexity of each iteration to be linear in the model rank $k$ when the rank of the gradient matrix $r$ is one.

Given rank-one gradients ($r = 1$), the most computationally demanding step in Algorithm 2 is the computation of the pseudo-inverse of the matrices $A$ and $B$, taking $O(nk^2)$ and $O(mk^2)$ operations. All other operations are $O(\max(n, m)k)$ at most. For $r = 1$ the outputs $Z_1$ and $Z_2$ become rank-one updates of the input matrices $A$ and $B$. This enables us to keep the pseudo-inverses $A^\dagger$ and $B^\dagger$ from the previous round, and perform a rank-one update to them, following a procedure developed by [9].

This procedure is similar to the better known Sherman-Morrison formula for the inverse of a rank-one perturbed matrix, and its computational complexity for an $n \times k$ matrix is $O(nk)$ operations. Using that procedure, we derive our final algorithm, *Loreta-1*, the rank-one Riemannian stochastic gradient descent. Its overall time and space complexity are both $O((n+m)k)$ per gradient step.

The memory requirement of Loreta-1 is about $4nk$ (assuming $m = n$), since it receives four input matrices of size $nk$ ($A, B, A^{\dagger}, B^{\dagger}$) and assuming it can compute the four outputs ($Z_1, Z_2, Z_1^{\dagger}, Z_2^{\dagger}$), in-place while destroying previously computed terms.

## 4 Related work

A recent summary of many advances in the field of optimization on manifolds is given in [4]. More specific to the field of low rank matrix manifolds, some work has been done on the general problem of optimization with low rank positive semi-definite (PSD) matrices. These include [10] and [6]; the latter introduced the retraction for PSD matrices which we extended here to general low-rank matrices. The problem of minimizing a convex function over the set of low rank matrices, was addressed by several authors, including [11], and [12] which also considers additional affine constraints, and its connection to recent advances in compresses sensing. The main tools used in these works are the trace norm (sum of singular values) and semi-definite programming. See also [2].

More closely related to the current paper are the works by Kulis et al. [13] and Meka et al. [14]. The first deals with learning low rank PSD matrices, and uses the rank-preserving log-det divergence and clever factorization and optimization in order to derive an update rule with runtime complexity of $O(nk^2)$ for an $n \times n$ matrix of rank $k$. The second uses online learning in order to find a minimal rank square matrix under approximate affine constraints. The algorithm does not directly allow a factorized representation, and depends crucially on an "oracle" component, which typically requires to compute an SVD. Multi-class ranking with a large number of features was studied in [3].

## 5 Experiments

We tested Loreta-1 in two learning tasks: learning a similarity measure between pairs of text documents using the 20-newsgroups data collected by [15], and learning to rank image label annotations based on a multi-label annotated set, using the *imagenet* dataset [16].[2]

### 5.1 Learning similarity on the 20 Newsgroups data set

In our first set of experiments, we looked at the problem of learning a similarity measure between pairs of text documents. Similarity learning is a well studied problem, closely related to metric learning (see [17] for a review). It has numerous applications in information retrieval such as *query by example*, and finding related content on the web.

One approach to learn pairwise relations is to measure the similarity of two documents $\mathbf{p}, \mathbf{q} \in \mathbb{R}^n$ using a bilinear form $S_W(\mathbf{p}, \mathbf{q}) = \mathbf{p}^T W \mathbf{q}$ parametrized by a model $W \in \mathbb{R}^{n \times n}$. Such models can be learned using standard online methods [8], and were shown to achieve high precision. Unfortunately, since the number of parameters grows as $n^2$, storing the matrix $W$ in memory is only feasible for limited feature dimensionality. To handle larger vocabularies, like those containing all textual terms found in a corpus, a common approach is to pre-select a subset of the features and train a model over the low dimensional data. However, such preprocessing may remove crucial signals in the data even if features are selected in a discriminative way.

To overcome this difficulty, we used Loreta-1 to learn a rank-$k$ parametrization of the model $W$, which can be factorized as $W = AB^T$, where $A, B \in \mathbb{R}^{n \times k}$. In each of our experiments, we selected a subset of $n$ features, and trained a rank $k$ model. We varied the number of features $n$ and the rank of the matrix $k$ so as to use a fixed amount of memory. For example, we used a rank-10 model with $50K$ features, and a rank-50 model with $10K$ features.

**Similarity learning with Loreta-1.** We use an online procedure similar to that in [7, 8]. At each round, three instances are sampled: a query document $\mathbf{q}$, and two documents $\mathbf{p}_1$ and $\mathbf{p}_2$ such that $\mathbf{p}_1$ is known to be more similar to $\mathbf{q}$ that $\mathbf{p}_2$. We wish that the model assigns a higher similarity score to the pair $(\mathbf{q}, \mathbf{p}_1)$ than the pair $(\mathbf{q}, \mathbf{p}_2)$, hence use the online ranking hinge loss defined as $l_W(\mathbf{q}, \mathbf{p}_1, \mathbf{p}_2) = [1 - S_W(\mathbf{q}, \mathbf{p}_1) + S_W(\mathbf{q}, \mathbf{p}_2)]_+$.

**Data preprocessing and feature selection.** We used the 20 newsgroups data set (people.csail.mit.edu/jrennie/20Newsgroups), containing 20 classes with approximately 1000 documents each. We removed stop words but did not apply stemming. We selected features that conveyed high information about the identity of the class (over the training set) using the *infogain* criterion [18]. The selected features were normalized using *tf-idf*, and then represented each document as a bag of words. Two documents were considered similar if they shared the same class label.

**Experimental procedure and evaluation protocol.** The 20 newsgroups site proposes a split of the data into train and test sets. We repeated splitting 5 times based on the sizes of the proposed splits (a train / test ratio of 65% / 35%). We evaluated the learned similarity measures using a ranking criterion. We view every document $\mathbf{q}$ in the test set as a query, and rank the remaining test documents $\mathbf{p}$ by their similarity scores $\mathbf{q}^T W \mathbf{p}$. We then compute the precision (fraction of positives) at the top $r$ ranked documents. We further compute the *mean average precision* (mAP), a widely used measure in the information retrieval community, which averages over all values of $r$.

**Comparisons.** We compared Loreta with the following approaches. **(1) A direct gradient descent (GD)** similar to [3]. The model is represented as a product of two matrices $\hat{W} = AB^T$. Stochastic gradient descent steps are computed over the factors $A$ and $B$, for the same loss used by Loreta $l_W(\mathbf{q}, \mathbf{p}_1, \mathbf{p}_2)$. The step size $\eta$ was selected using cross validation. The GD steps are: $A_{new} = A + \eta \mathbf{q}(\mathbf{p}_1 - \mathbf{p}_2)^T B$, and $B_{new} = A + \eta(\mathbf{p}_1 - \mathbf{p}_2)\mathbf{q}^T A$. **(2) Iterative Passive-Aggressive (PA)**. We found the above GD procedure to be very unstable, often causing the models to diverge. We therefore used a related online algorithm from the family of passive-aggressive algorithms [19]. We iteratively optimize over $A$ given a fixed $B$ and vice versa. The optimization is a tradeoff between minimizing the loss $l_W$, and limiting how much the models change at each iteration. The steps size for updating $A$ is computed to be $\eta_A = \max(\frac{l_W(\mathbf{q},\mathbf{p}_1,\mathbf{p}_2)}{\|\mathbf{q}\|^2 \|B^T(\mathbf{p}_1-\mathbf{p}_2)\|^2}, C)$, and $\eta_B = \max(\frac{l_W(\mathbf{q},\mathbf{p}_1,\mathbf{p}_2)}{\|(\mathbf{p}_1-\mathbf{p}_2)\|^2 \|A^T \mathbf{q}\|^2}, C)$. $C$ is a predefined parameter controlling the maximum magnitude of the step size. This procedure is numerically more stable because of the normalization by the norms of the matrices multiplied by the gradient factors. **(3) Naive Passive-Aggressive (PA v2)** This method is similar to the iterative PA above, with the step size computed as with unfactored matrices $\eta = \max(\frac{l_W(\mathbf{q},\mathbf{p}_1,\mathbf{p}_2)}{\|q\|^2 \|(\mathbf{p}_1-\mathbf{p}_2)\|^2}, C)$. **(4) Full rank similarity learning models.** We compared with two online metric learning methods, LEGO [20] and OASIS [8]. Both algorithms learn a full (non-factorized) model, and were run with $n = 1000$, in order to be consistent with the memory constraint of Loreta-1. We have not compared with batch approaches such as [13]

Figure 2b shows the mean average precision obtained with the three measures. Loreta outperforms the PA approach across all ranks. More importantly, learning a low rank model of rank 30, using the best 16660 features, is significantly more precise than learning a much fuller model of rank 100 and 5000 features. The intuition is that Loreta can be viewed as adaptively learning a linear projection of the data into low dimensional space, which is tailored to the pairwise similarity task.

## 5.2 Image multilabel ranking

Our second set of experiments tackled the problem of learning to rank labels for images taken from a large number of classes ($L = 1661$) with multiple labels per image.

In our approach, we learn a linear classifier over $n$ features per each label $c \in \mathcal{C} = \{1, \ldots, L\}$, and stack all models together to a single matrix $W \in \mathbb{R}^{L \times n}$. At test time, given an image $\mathbf{p} \in \mathbb{R}^n$, the product $W\mathbf{p}$ provides scores for every label per that image $\mathbf{p}$. Given a ground truth labeling, a good model would rank the true labels higher than the false ones. Each row of the matrix model can be thought of as a sub-model for the corresponding label. Imposing a low rank constraint on the model implies that these sub-models are linear combinations of a smaller number of latent models.

**Online learning of label rankings with Loreta-1.** At each iteration, an image $\mathbf{p}$ is sampled, and using the current model $W$ the scores for all its labels were computed, $W\mathbf{p}$. These scores are compared with the ground truth labeling $\mathbf{y} = \{y_1, \ldots, y_r\} \subset \mathcal{C}$. The learner suffers a multilabel multiclass hinge loss as follows. Let $\bar{y} = \text{argmax}_{s \notin \mathbf{y}}(W\mathbf{p})_s$, be the negative label which obtained the highest score, where $(W\mathbf{p})_s$ is the $s^{th}$ component of the score vector $(W\mathbf{p})$. The loss is then $\mathcal{L}(W, \mathbf{p}, \mathbf{y}) = \sum_{i=1}^{r} [(W\mathbf{p})_{\bar{y}} - (W\mathbf{p})_{y_i} + 1]_+$. We then used the subgradient $G$ of this loss for Loreta: for the set of indices $i_1, i_2, \ldots i_d \subset \mathbf{y}$ which incurred a non zero hinge loss, the $i_j$ row of $G$ is $\mathbf{p}$, and for the row $\bar{y}$ we set $G$ to be $-d \cdot \mathbf{p}$. The matrix $G$ is rank one, unless no loss was suffered in which case it is 0. .

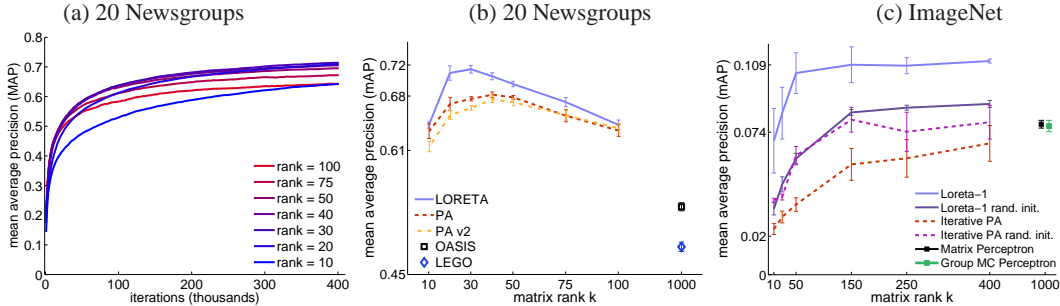

Figure 2: **(a)** Mean average precision (mAP) over 20 newsgroups test set as traced along Loreta learning for various ranks. Curve values are averages over 5 train-test splits. **(b)** mAP of different models with varying rank. For each rank, a different number of features was selected using an information gain criterion, such that the total memory requirement is kept fixed (number of features × rank is constant). 50000 features were used for rank = 10. LEGO and OASIS were trained with the same memory (using 1000 features and rank=1000). Error bars denote the standard error of the mean over 5 train-test splits (s.e.m.). **(c)** ImageNet data. mAP as a function of the rank $k$. Curves are means over three train-test splits. Error bars denote the standard error of the mean (s.e.m.). All hyper parameters were selected using cross validation. Models were initialized either with $k$ ones along the diagonal, or as a product of rank-$k$ matrices with random normal entries (denoted rand. init.).

**Data set and preprocessing.** We used a subset of the ImageNet 2010 Challenge (www.imagenet.org/challenges/LSVRC/2010/) containing images labeled with respect to the Word-Net hierarchy. Each image was manually labeled with a single class label (for a total of 1000 classes). We added labels for each image, using classes along the path to the root of the hierarchy (adding 676 classes in total). We discarded ancestor labels covering more than 10% of the images, leaving 1661 labels (5.3 labels per image on average). We used ImageNets bag of words representation, based on vector quantizing SIFT features with a vocabulary of 1000 words, followed by *tf-idf* normalization.

**Experimental procedure and evaluation protocol.** We split the data into 30 training and 20 testing images per every base level label. The quality of the learned label ranking, was evaluated using the *mean average precision* (mAP) criterion mentioned above.

**Comparisons.** We compared the performance of Loreta on this task with three other approaches: **(1) PA: Iterative Passive-Aggressive** as described above. **(2) Matrix Perceptron**: a full rank conservative gradient descent **(3) Group Multi-Class Perceptron** a mixed (2,1) norm online mirror descent algorithm [21]. Loreta and PA were run using a range of different model ranks. For all three methods the step size (or C parameter for the PA) was chosen by 5-fold validation on the test set.

Figure Fig. 2c plots the mAP precision of Loreta and PA for different model ranks, while showing on the right the mAP of the full rank 1000 gradient descent and $(2, 1)$ norm algorithms. Loreta significantly improves over all other methods across all ranks.

## 6 Discussion

We presented Loreta, an algorithm which learns a low-rank matrix based on stochastic Riemannian gradient descent and efficient retraction to the manifold of low-rank matrices. Loreta achieves superior precision in a task of learning similarity in high dimensional feature spaces, and in multi-label annotation, where it scales well with the number of classes.

Loreta yields a factorized representation of the low rank matrix. For classification, it can be viewed as learning two matrix components: one that projects the high dimensional data into a low dimension, and a second that learns to classify in the low dimension. It may become useful in the future for exploring high dimensional data, or extract relations between large number of classes.

### Acknowledgments

This work was supported by the Israel Science Foundation (ISF) and by the European Union under the DIRAC integrated project IST-027787.

## Footnotes

[2]Matlab code for Loreta-1 can be provided upon request.

# References

[1] B.K. Natarajan. Sparse approximate solutions to linear systems. *SIAM journal on computing*, 24(2):227–234, 1995.

[2] M. Fazel, H. Hindi, and S. Boyd. Rank minimization and applications in system theory. In *Proceedings of the 2004 American Control Conference*, pages 3273–3278. IEEE, 2005.

[3] B. Bai, J. Weston, R. Collobert, and D. Grangier. Supervised semantic indexing. *Advances in Information Retrieval*, pages 761–765, 2009.

[4] P.A. Absil, R. Mahony, and R. Sepulchre. *Optimization Algorithms on Matrix Manifolds*. Princeton Univ Press, 2008.

[5] P.-A. Absil and Jérôme Malick. Projection-like retractions on matrix manifolds. Technical Report UCL-INMA-2010.038, Department of Mathematical Engineering, Université catholique de Louvain, July 2010.

[6] B. Vandereycken and S. Vandewalle. A Riemannian optimization approach for computing low-rank solutions of Lyapunov equations. *SIAM Journal on Matrix Analysis and Applications*, 31:2553, 2010.

[7] D. Grangier D. and S. Bengio. A discriminative kernel-based model to rank images from text queries. *IEEE Transactions on Pattern Analysis and Machine Intelligence*, 30:1371–1384, 2008.

[8] G. Chechik, V. Sharma, U. Shalit, and S. Bengio. Large scale online learning of image similarity through ranking. *Journal of Machine Learning Research*, 11:1109–1135, 2010.

[9] C.D. Meyer. Generalized inversion of modified matrices. *SIAM Journal on Applied Mathematics*, 24(3):315–323, 1973.

[10] M. Journee, F. Bach, PA Absil, and R. Sepulchre. Low-Rank Optimization on the Cone of Positive Semidefinite Matrices. *SIAM Journal on Optimization*, 20:2327–2351, 2010.

[11] M. Fazel. *Matrix rank minimization with applications*. PhD thesis, Electrical Engineering Department, Stanford University, 2002.

[12] Benjamin Recht, Maryam Fazel, and Pablo A. Parrilo. Guaranteed minimum-rank solutions of linear matrix equations via nuclear norm minimization. *SIAM Review*, 52(3):471–501, 2010.

[13] B. Kulis, M.A. Sustik, and I.S. Dhillon. Low-rank kernel learning with bregman matrix divergences. *The Journal of Machine Learning Research*, 10:341–376, 2009.

[14] R. Meka, P. Jain, C. Caramanis, and I.S. Dhillon. Rank minimization via online learning. In *Proceedings of the 25th International Conference on Machine learning*, pages 656–663, 2008.

[15] K. Lang. Learning to filter netnews. In *Proceeding of the 12th Internation Conference on Machine Learning*, pages 331–339, 1995.

[16] J. Deng, W. Dong, R. Socher, L.J. Li, K. Li, and L. Fei-Fei. ImageNet: a large-scale hierarchical image database. In *Proceedings of the 22nd IEEE Conference on Computer Vision and Pattern Recognition*, 2009.

[17] L. Yang. An overview of distance metric learning. Technical report, School of Computer Science, Carnegie Mellon University, 2007.

[18] Y. Yang and J.O. Pedersen. A comparative study on feature selection in text categorization. In *Proceedings of the 14th International Conference on Machine learning*, pages 412–420, 1997.

[19] K. Crammer, O. Dekel, J. Keshet, S. Shalev-Shwartz, and Y. Singer. Online passive-aggressive algorithms. *Journal of Machine Learning Research*, 7:551–585, 2006.

[20] P. Jain, B. Kulis, I.S. Dhillon, and K. Grauman. Online metric learning and fast similarity search. *Advances in Neural Information Processing Systems*, pages 761–768, 2008.

[21] Sham M. Kakade, Shai Shalev-Shwartz, and Ambuj Tewari. Regularization techniques for learning with matrices, 2010. preprint.

